# Contraction Properties of VLSI Cooperative Competitive Neural Networks of Spiking Neurons

**Emre Neftci**[1], **Elisabetta Chicca**[1], **Giacomo Indiveri**[1], **Jean-Jacques Slotine**[2], **Rodney Douglas**[1]
[1]Institute of Neuroinformatics, UNI|ETH, Zurich
[2]Nonlinear Systems Laboratory, MIT, Cambridge, Massachusetts, 02139
emre@ini.phys.ethz.ch

## Abstract

A non–linear dynamic system is called *contracting* if initial conditions are forgotten exponentially fast, so that all trajectories converge to a single trajectory. We use contraction theory to derive an upper bound for the strength of recurrent connections that guarantees contraction for complex neural networks. Specifically, we apply this theory to a special class of recurrent networks, often called Cooperative Competitive Networks (CCNs), which are an abstract representation of the cooperative-competitive connectivity observed in cortex. This specific type of network is believed to play a major role in shaping cortical responses and selecting the relevant signal among distractors and noise. In this paper, we analyze contraction of combined CCNs of linear threshold units and verify the results of our analysis in a hybrid analog/digital VLSI CCN comprising spiking neurons and dynamic synapses.

## 1 Introduction

Cortical neural networks are characterized by a large degree of recurrent excitatory connectivity, and local inhibitory connections. This type of connectivity among neurons is remarkably similar, across all areas in the cortex [1]. It has been argued that a good candidate model for a canonical micro-circuit, potentially used as a general purpose cortical computational unit in the cortices, is the soft Winner-Take-All (WTA) circuit [1], or the more general class of Cooperative Competitive Networks [2] (CCN). A CCN is a set of interacting neurons, in which cooperation is achieved by local recurrent excitatory connections and competition is achieved via a group of inhibitory neurons, driven by the excitatory neurons and inhibiting them (see Figure 1). As a result, CCNs perform both common linear operations as well as complex non–linear operations. The linear operations include analog gain (linear amplification of the feed–forward input, mediated by the recurrent excitation and/or common mode input), and locus invariance [3]. The non–linear operations include non–linear selection or soft winner–take–all (WTA) behavior [2, 4, 5], signal restoration [4, 6], and multi–stability [2, 5]. CCN networks can be modeled using linear threshold units, as well as recurrent networks of spiking neurons. The latter can be efficiently implemented in silicon using Integrate–and–Fire (I&F) neurons and dynamic synapses [7]. In this work we use a prototype VLSI CCN device, comprising 128 low power I&F neurons [8] and 4096 dynamic synapses [9] that operate in real-time, in a massively parallel fashion. The main goal of this paper is to address the open question of how to determine network parameters, such as the strength of recurrent excitatory couplings or global inhibitory couplings, to create well–behaving complex networks composed of combinations of neural computational modules (such as CCNs) as depicted in Figure 1. The theoretical foundations used to address these problems are based on *contraction theory* [10]. By applying this theory to CCN models of linear threshold units and to combinations of them we find upper bounds to contraction conditions. We then test the theoretical results on the VLSI CCN of spiking neurons, and on a combination of two mutually coupled CCNs. We show how the experimental data presented are consistent with the theoretical predictions.

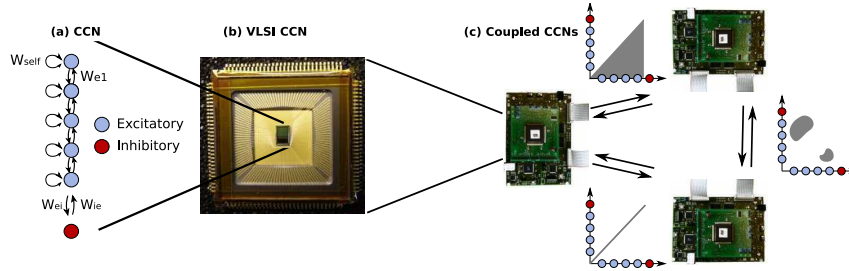

Figure 1: *CCNs and combinations of CCNs*. (a) A CCN consisting of a population of nearest neighbor connected excitatory neurons (blue) receiving external input and an inhibitory neuron which receives input from all the excitatory neurons and inhibits them back (red). (b) Photo of the VLSI CCN Chip comprising I&F neurons. (c) Three coupled CCNs, showing examples of connectivity patterns between them.

## 2   CCN of linear threshold units

Neural network models of linear threshold units (LTUs) ignore many of the non–linear processes that occur at the synaptic level and contain, by definition, no information about spike timing. However networks of LTUs can functionally behave as networks of I&F neurons in a wide variety of cases [11]. Similarly boundary conditions found for LTU networks can be often applied also to their I&F neuron network counterparts. For this reason, we start by analyzing a network of LTUs whose structure is analogous to the one of the VLSI CCN of I&F neurons, and derive sufficient boundary conditions for contraction.

If we consider a CCN of recurrently connected LTUs according to a weight matrix $\mathbf{W}$, as shown on Figure 1, we can express the network dynamics as:

$$\tau_i \frac{\mathrm{d}}{\mathrm{d}t}x_i = -x_i + g((\mathbf{W}\mathbf{x})_i + b_i) \quad \forall i = 1,...,N \tag{1}$$

Where $N$ is the total number of neurons in the system, the function $g(x) = \max(x,0)$ is a half–wave rectification non–linearity to ensure that $\mathbf{x} \equiv (x_1,...,x_N)^\top$ remains positive, $b_i$ are the external inputs applied to the neurons and $\tau_i$ are the time constants of the neurons. We assume that neurons of each type (i.e. excitatory or inhibitory) have identical dynamics: we denote the time constant of excitatory neurons with $\tau_{ex}$ and the one of inhibitory neurons with $\tau_{ih}$. Throughout the paper, we will use the following notation for the weights: $w_s$ for self excitation, $w_{e1}, w_{e2}$ for $1^{st}$ and $2^{nd}$ nearest neighbor excitation respectively, and $w_{ie}, w_{ei}$ for inhibitory to excitatory neuron and vice versa. The $\mathbf{W}$ matrix has the following shape:

$$\mathbf{W} = \begin{pmatrix} w_{self} & w_1 & w_2 & 0 & w_2 & w_1 & -w_{ei} \\ & & & \ddots & & & \vdots \\ w_1 & w_2 & 0 & w_2 & w_1 & w_{self} & -w_{ei} \\ w_{ie} & w_{ie} & w_{ie} & w_{ie} & w_{ie} & w_{ie} & 0 \end{pmatrix} \tag{2}$$

A CCN can be used to implement a WTA computation. Depending on the strength of the connections, a CCN can implement a *Hard* (HWTA) or *Soft* (SWTA) WTA. A HWTA implements a *max* operation or selection mechanism: only the neuron receiving the strongest input can be active and all other neurons are suppressed by global inhibition. A SWTA implements more complex operation such as non–linear selection, signal restoration, and multi–stability: one or several groups of neurons can be active at the same time, neurons belonging to the same group cooperate through local excitation, different groups compete through global inhibition. The activity of the 'winning' group of neurons can be amplified while other groups are suppressed. Depending on the strength of inhibitory and excitatory couplings different regimes are observed. Specifically, in Sec. 4 we compare a weakly coupled configuration, which guarantees contraction, with a strongly coupled configuration in which the output of the network depends on the input and the history, showing hysteretic (non–contracting) behaviors in which the selected 'winning' group has advantages over other group of neurons because of the recurrent excitation.

# 3 Contraction theory applied to CCNs of linear threshold units

## 3.1 Contraction of a single network

A formal analysis of contraction theory applied to non–linear systems has been described in [10,12]. Here we present an overview of the theory applied to the system of Eq. (1).

In a *contracting* system, all the trajectories converge to a single trajectory exponentially fast independent of the initial conditions. In particular, if the system has a steady state solution then, by definition, the state will contract and converge to that solution exponentially fast. Formally, the system is contracting if $\frac{d}{dt} \| \delta \mathbf{x} \|$ is uniformly negative (i.e. negative in the entire state space) where $\delta \mathbf{x}$ corresponds to the distance between two neighboring trajectories at a given time. In fact, by path integration, we have $\frac{d}{dt} \int_{P_2}^{P_1} \| \delta \mathbf{x} \| < 0$ where $P_1$ and $P_2$ are two points of state space (non-necessarily neighboring). This leads to the following theorem:

*Consider a system whose dynamics is given by the differential equations $\frac{d}{dt}\mathbf{x} = \mathbf{f}(\mathbf{x},t)$. The system is said to be* contracting *if all its trajectories converge exponentially to a single trajectory. A sufficient condition is that the* symmetric part *of the Jacobian $\mathbf{J} = \frac{\partial}{\partial \mathbf{x}}\mathbf{f}$ is uniformly negative definite. This condition can be written more explicitly as*

$$\exists \beta > 0, \forall \mathbf{x}, \forall t \geq 0 \quad \mathbf{J}_s \equiv \frac{1}{2}(\mathbf{J} + \mathbf{J}^\top) \leq -\beta \mathbb{I}$$

*where $\mathbb{I}$ is the identity matrix and $\mathbf{J}_s$ is the symmetric part of $\mathbf{J}$ It is equivalent to $\mathbf{J}_s$ having all its eigenvalues uniformly negative [13].*

We can define more generally a local coordinate transformation $\delta \mathbf{z} = \Theta \delta \mathbf{x}$, where $\Theta(\mathbf{x},t)$ is a square matrix, such that $\mathbf{M}(\mathbf{x},t) = \Theta^T \Theta$ is a uniformly positive definite, symmetric and continuously differentiable metric. Note that the coordinate system $\mathbf{z}(\mathbf{x},t)$ does not need to exist, and will not in the general case, but $\delta \mathbf{z}$ and $\delta \mathbf{z}^\top \delta \mathbf{z}$ can always be defined [14]. Then, in this metric one can compute the *generalized Jacobian* $\mathbf{F} = (\frac{d}{dt}\Theta + \Theta \mathbf{J})\Theta^{-1}$. If the symmetric part of the generalized Jacobian, $\mathbf{F_s}$, is negative definite then the system is contracting. In a suitable metric it has been shown that this condition becomes sufficient and necessary [10]. In particular, if $\Theta$ is constant, $\mathbf{F}_s$ is negative definite if and only if $(\mathbf{MF})_s$ is negative definite. In fact, as $\mathbf{F}_s = (\Theta^{-1})^T (\mathbf{MJ})_s \Theta^{-1}$, then the condition $v^T \mathbf{F}_s v < 0 \quad \forall v \in \mathbb{R}^N$ (negative definite matrix) is equivalent to $(v^T (\Theta^{-1})^T)(\mathbf{MJ})_s (\Theta^{-1} v) < 0 \quad \forall \Theta^{-1} v \in \mathbb{R}^N$. Consequently, we can always choose a constant $\mathbf{M}$ to simplify our equations.

Let us now see under which conditions the system defined by Eq. (1) is contracting. Except for the rectification non–linearity, the full system is a linear time–invariant (LTI) system, and it has a fixed point [15]. A common alternative to the half-wave rectification function is the sigmoid, in which case the Jacobian becomes differentiable. If we define $f_i(\mathbf{x},t)$ as

$$f_i(\mathbf{x},t) \equiv \frac{d}{dt}x_i = -\frac{1}{\tau_i}x_i + \frac{1}{\tau_i}g((\mathbf{Wx})_i + b_i) \tag{3}$$

then the Jacobian matrix is given by $\mathbf{J}_{ij} = \frac{\partial}{\partial x_j}f_i(\mathbf{x},t) = -\frac{1}{\tau_i}\delta_{ij} + \frac{1}{\tau_i}g'(\mathbf{y}_i)w_{ij}$, where $\mathbf{y}_i = (\mathbf{Wx})_i + b$ and $\tau_i$ is the time constant of neuron $i$, with $\tau_i = \tau_{ex}$ for the excitatory neurons and $\tau_i = \tau_{ih}$ for the inhibitory ones. We assume that the $w_{ei}$ and $w_{ie}$ weights are not zero so we can use the constant metric:

$$\mathbf{M} = \begin{pmatrix} \tau_{ex} & 0 & 0 \\ \vdots & \ddots & \vdots \\ 0 & \cdots & \frac{w_{ei}}{w_{ie}}\tau_{ih} \end{pmatrix} \tag{4}$$

which is positive definite. With this metric, $\mathbf{MJ}$ can be written $\mathbf{MJ} = -\mathbb{I} + \mathbf{DK}$, where $\mathbf{D}_{ij} = g'(\mathbf{y}_i)\delta_{ij}$, and $\mathbf{K}$ is similar to $\mathbf{W}$ but with $w_{ei}$ in place of $w_{ie}$. Since $g$ is sigmoidal (and thus it and its derivative are both bounded), we can then use the method proposed in [16] to determine a sufficient condition for contraction. This leads to a condition of the form $\lambda_{max} < 0$, where

$$\lambda_{max} = 2w_{e1} + 2w_{e2} + w_s - 1 \tag{5}$$

A graphical representation of the boundaries defined by this contraction condition is provided in Figure 2. The term $|\lambda_{max}|$ is called *the contraction rate* with respect to metric $\mathbf{M}$. It is of particular interest because it is a lower bound for the rate at which the system converges to its solution in that metric.

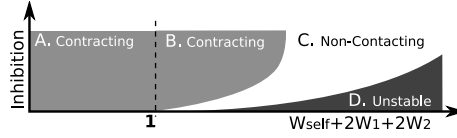

Figure 2: *Qualitative phase diagram for a single CCN of LTUs.* We show here the possible regimes of the given in Eq. (1) as a function of excitation and inhibition. In the region **D** the rates would grow without bounds if there were no refractory period for the neurons. We see that a system which is unstable without inhibition cannot be in region **A** (*i.e.* within the boundaries of Eq. (5)). Note, however, that we do not quantitatively know the boundaries between **B** and **C** and between **C** and **D**

## 3.2 Contraction of feed–back combined CCNs

One of the powerful features of contraction theory is the following: if a complex system is composed of coupled (feed–forward and feed–back) subsystems that are individually contracting, then it is possible to find a sufficient condition for contraction without computing the system's full Jacobian. In addition it is possible to compute a lower bound for the full system's contraction rate. Let $\mathbf{F}_s$ be the symmetric part of the Jacobian of two bi–directionally coupled subsystems, with symmetric feed–back couplings. Then $\mathbf{F}_s$ can be written with four blocks of matrices:

$$\mathbf{F}_s = \begin{bmatrix} \mathbf{F}_{1s} & \mathbf{G} \\ \mathbf{G}^\top & \mathbf{F}_{2s} \end{bmatrix} \tag{6}$$

where $\mathbf{F}_{1s}$ and $\mathbf{F}_{2s}$ refer to the Jacobian of the individual, decoupled subsystems, while $\mathbf{G}$ and $\mathbf{G}^\top$ are the feed–back coupling components. If we assume both subsystems are contracting, then a sufficient condition for contraction of the overall system is given by [17]:

$$|\lambda_{max}(\mathbf{F}_{1s})| \, |\lambda_{max}(\mathbf{F}_{2s})| > \sigma^2(\mathbf{G}) \quad \forall t > 0, \, uniformly \tag{7}$$

where $|\lambda_{max}(\cdot)|$ is the contraction rate with respect to the used metric and $\sigma(\mathbf{G})$ is the largest eigenvalue of $\mathbf{G}^\top \mathbf{G}$. By the eigenvalue interlacing theorem [13] we have that the contraction rate of the combined system is given by $\lambda_{max}(\mathbf{F}_s) \leq \min_i \lambda(\mathbf{F}_{is}) \quad i = 1, 2$.

For the specific example of a combined system comprising two identical subsystems coupled by a uniform coupling matrix $\mathbf{G} = w_{fb} \cdot \mathbb{I}$ we have $\sigma^2(\mathbf{G}) = w_{fb}^2$. The combined system is contracting if:

$$|w_{fb}| < \lambda_{max} \tag{8}$$

The results obtained with this analysis can be generalized to more than two combined subsystems, and with different types of coupling matrices [17]. Note that in a feed–forward or a negative–feedback case (i.e. at least one of the 'G–blocks' in the non–symmetric form is negative semidefinite), the system is automatically contracting provided that both subsystems are contracting. Given this, the condition for contraction of the combined system described by Eq. (8) becomes: $w_{fb} < \lambda_{max}$. Note that the contraction rate is an observable quantity, therefore one can build a contracting system consisting of an arbitrary number of CCNs as follows: 1. Determine the contraction rate of two CCNs by using Eq. (5) or by measuring it. 2. Use Eq. (7) to set the weight of the relation. Compute the upper bound to the contraction rate of the combined system as explained above. 3. Repeat the procedure for a new CCN and the combined one.

## 4 Contraction in a VLSI CCN of spiking neurons

The VLSI device used in this work implements a CCN of spiking neurons using an array of low–power I&F neurons with dynamic synapses [8, 18]. The chip has been fabricated using a standard AMS $0.35\mu m$ CMOS process, and covers an area of about $10\,mm^2$. It contains 124 excitatory neurons with self, $1^{st}$, $2^{nd}$, $3^{rd}$ nearest–neighbor recurrent excitatory connections and 4 inhibitory neurons (all–to–all bi–directionally connected to the excitatory neurons). Each neuron receives input currents from a row of 32 afferent plastic synapses that use the Address Event Representation (AER) to receive spikes. The spiking activity of the neurons is also encoded using the AER. In this representation input and output spikes are real–time asynchronous digital events that carry analog information in their temporal structure. We can interface the chip to a workstation, for prototyping

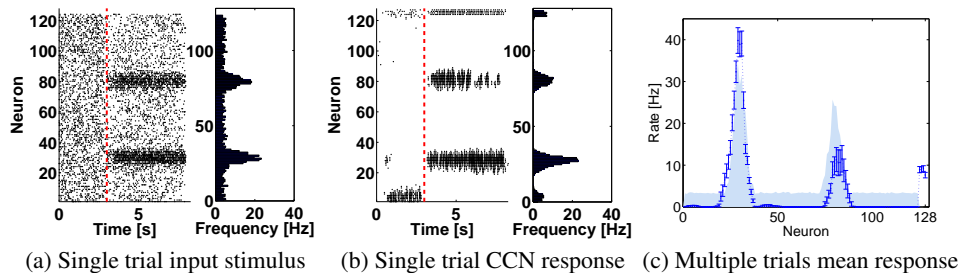

(a) Single trial input stimulus    (b) Single trial CCN response    (c) Multiple trials mean response

Figure 3: *Contraction of a single VLSI CCN*. (a) A raster plot of the input stimulus(left) and the mean firing rates(right): the membrane potential of the I&F neurons are set to a random initial state by stimulating them with uncorrelated Poisson spike trains of constant mean frequency (up to the dashed line). Then the network is stimulated with 2 Gaussian bumps of different amplitude centered at Neuron 30 and Neuron 80, while, all the neurons received a constant level of uncorrelated input during the whole trial. (b) The response of the CCN to the stimulus presented in (a). (c) Mean responses of 100 trials, calculated after the red dashed line with error bars. The shaded area represents the mean input stimulus presented throughout the experiment. The system selects the largest input and suppresses the noise and the smaller bump, irrespective of initial conditions and noise. Neurons 124 to 128 are inhibitory neurons and do not receive external input.

experiments using a dedicated PCI–AER board [19]. This board allows us to stimulate the synapses on the chip (*e.g.* with synthetic trains of spikes), monitor the activity of the I&F neurons, and map events from one neuron to a synapse belonging to a neuron on the same chip and/or on a different chip. An analysis of the dynamics of our VLSI I&F neurons can be found in [20] and although the leakage term in our implemented neurons is constant, it has been shown that such neurons exhibit responses qualitatively similar to standard linear I&F neurons [20].

A steady state solution is easily computable for a network of linear threshold units [5, 21]: it is a fixed point in state space, *i.e.* a set of activities for the neurons. In a VLSI network of I&F neurons the steady state will be modified by mismatch and the activities will fluctuate due to external and microscopic perturbations (but remain in its vicinity if the system is contracting). To prove contraction experimentally in these types of networks, one would have to apply an input and test with all possible initial conditions. This is clearly not possible, but we can verify under which conditions the system is compatible with contraction by repeating the same experiment with different initial conditions (see Sec. 4.1) and under which conditions the system is not compatible with contraction by observing if system settles to different solutions when stimulated with different initial conditions (see Sec. 4.3).

## 4.1 Convergence to a steady state with a static stimulus

The VLSI CCN is stimulated by uncorrelated Poisson spike trains whose mean rates form two Gaussian–shaped bumps along the array of neurons, one with a smaller amplitude than the other superimposed to background noise (see Figure 3a). In a SWTA configuration, our CCNs should select and amplify the largest bump while suppressing the smaller one and the noise. We set the neurons into random initial conditions by stimulating them with uncorrelated Poisson spike trains with a spatially uniform and constant mean rate, before applying the real input stimulus (before the dashed line in Figure 3a ). Figure 3b shows the response of the CCN to this spike train, and Figure 3c is the response averaged over 100 trials. This experiment shows that regardless of the initial conditions, the final response of the CCN in an SWTA configuration is always the same (see the small error bars on Figure 3c), as we would expect from a contracting system.

## 4.2 Convergence with non–static stimulus and contraction rate

As the condition for contraction does not depend on the external input, it will also hold for time–varying inputs. For example an interesting input stimulus is a bump of activity moving along the array of neurons at a constant speed. In this case, the firing rates produced by the chip carry informa-

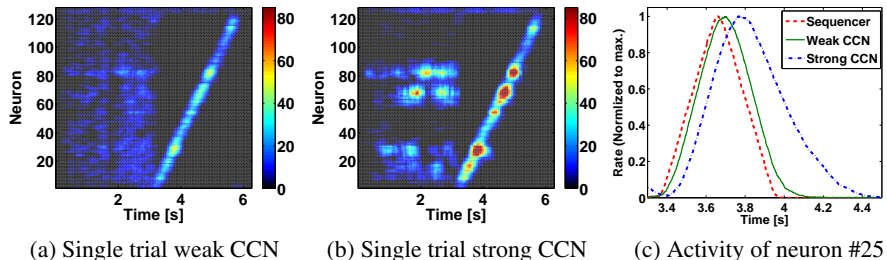

(a) Single trial weak CCN      (b) Single trial strong CCN      (c) Activity of neuron #25

Figure 4: *Contraction rate in VLSI CCNs using non–static stimulation*. The input changed from an initial stage, where all the neurons were randomly stimulated with constant mean frequencies (up to 3 s), to a second stage in which the moving stimulus (freshly generated from trial to trial) is applied. This stimulus consists of a bump of activity that is shifted from one neuron to the next. Panels (a) and (b) show trials for two different configurations (weak and strong) and the colors indicate the firing rates calculated with a 300 ms sliding time window. The panel (c) compares the mean rates of neuron #25 in the weakly coupled CCN (green), the strong CCN (blue) and the input (red), all normalized to their peak of activity and calculated over 50 trials. We see how the blue line is delayed compared to the red and green line: the stronger recurrent couplings reduces the contraction rate.

tion about the system's contraction rate. We measured the response of the chip to such a stimulus, for both strong and weak recurrent couplings (see Figure 4). The strong coupling case produces slower responses to the input than the weak coupling case, as expected from a system having a lower contraction rate (see Figure 4c). The system's condition for contraction does not depend on the individual neuron's time constants, although the *contraction rate* in the original metric does. This also applies to the non–static input case, where the system will converge to the expected solution, independently of the neurons time constants. Local mismatch effects in the VLSI chip lead to an effective weight matrix whose elements $w_{self}, w_1, w_2, w_{ie}$ are not identical throughout the array. This combined with the high gain of the strong coupling, and the variance produced by the input Poisson spike trains during the initial phase, explains the emergence of "pseudo-random" winners around neuron 30,60 and 80 in Figure 4b.

### 4.3   A non–contracting example

We expect a CCN to be non–contracting when the coupling is strong: in this condition the CCN exhibits a hysteretic behavior [22], so the position of the winner strongly depends on the network's initial conditions. Figure 5 illustrates this behavior with a CCN with very strong recurrent weights.

### 4.4   Contraction of combined systems

By using a multi-chip AER communication infrastructure [19] we can connect multiple chips together with arbitrary connectivity matrices (e.g. **G** in Sec. 3.2), and repeat experiments analogous to the ones of Sec. 4.1. Figure 6 shows the response of two CCNs, combined via a connectivity matrix as shown in Figure 6b, to three input bumps of activity in a contracting configuration.

## 5   Conclusion

We applied contraction theory to combined Cooperative Competitive Networks (CCN) of Linear Threshold Units (LTU) and determined sufficient conditions for contraction. We then tested the theoretical predictions on neuromorphic VLSI implementations of CCNs, by measuring their response to different types of stimuli with different random initial conditions. We used these results to determine parameter settings of single and combined networks of spiking neurons which make the system behave as a contracting one. Similarly, we verified experimentally that CCNs with strong recurrent couplings are not contracting as predicted by the theory.

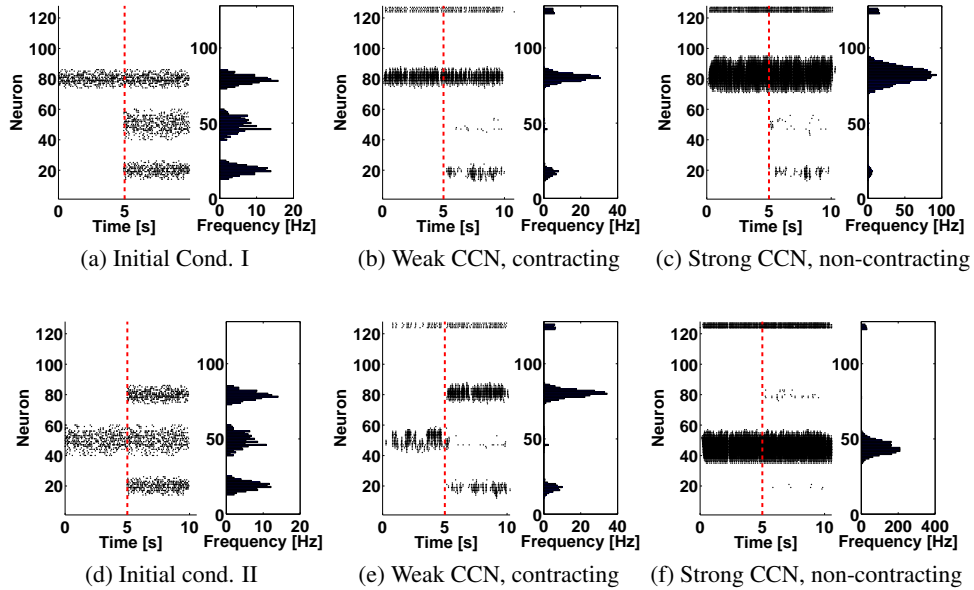

(a) Initial Cond. I     (b) Weak CCN, contracting     (c) Strong CCN, non-contracting

(d) Initial cond. II     (e) Weak CCN, contracting     (f) Strong CCN, non-contracting

Figure 5: *VLSI CCN in a non-contracting configuration.* We compare the CCN with very strong lateral recurrent excitation and low inhibition to a weakly coupled CCN. The figures present the raster plot and mean rates of the CCNs response (calculated after the dashed line) to the same stimuli starting from two different initial conditions. Panels (b) and (e) show the response of a contracting CCN, whereas panels (c) and (f) show that the system response depends on the initial conditions of (a) and (d). Therefore the the "Strong CCN" is non–contracting.

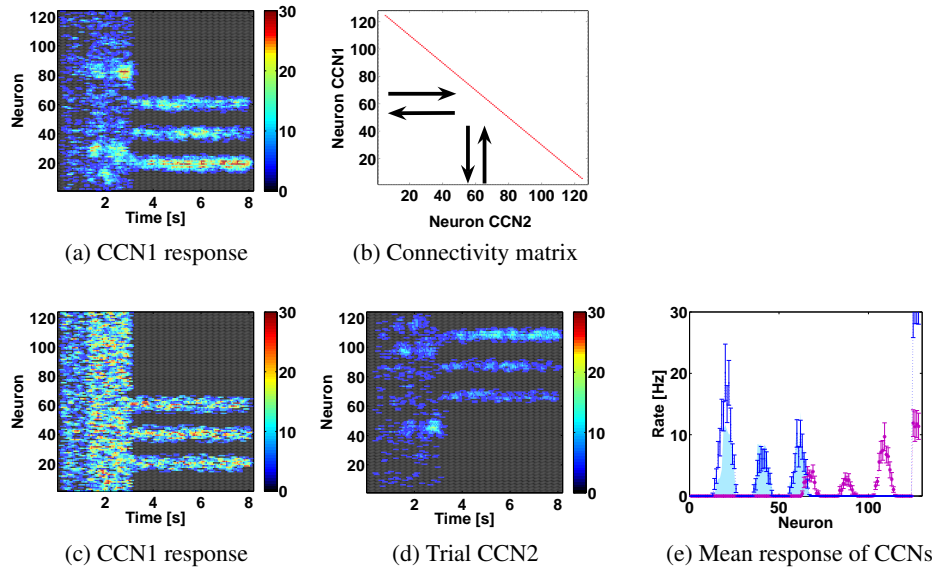

(a) CCN1 response     (b) Connectivity matrix

(c) CCN1 response     (d) Trial CCN2     (e) Mean response of CCNs

Figure 6: *Contraction in combined CCNs.* (a) and (d) Single trial responses of CCN1 and CCN2 to the input stimulus shown in (c); (b) Connectivity matrix that couples the two CCNs (inverted identity matrix); (e) Mean response of CCNs, averaged over 20 trials (data points) superimposed to average input frequencies (shaded area). The response of the coupled CCNs converged to the same mean solution, consistent with the hypothesis that the combined system is contracting.

## Acknowledgments

This work was supported by the DAISY (FP6-2005-015803) EU grant, and by the Swiss National Science Foundation under Grant PMPD2-110298/1. We thank P. Del Giudice and V. Dante (ISS), for original design of the PCI-AER board and A. Whatley for help with the software of the PCI-AER board.

## References

[1] R.J. Douglas and K.A.C. Martin. Neural circuits of the neocortex. *Annual Review of Neuroscience*, 27:419–51, 2004.

[2] S. Amari and M. A. Arbib. Competition and cooperation in neural nets. In J. Metzler, editor, *Systems Neuroscience*, pages 119–65. Academic Press, 1977.

[3] D. Hansel and H. Somplinsky. *Methods in Neuronal Modeling*, chapter Modeling Feature Selectivity in Local Cortical Circuits, pages 499–567. MIT Press, Cambridge, Massachusetts, 1998.

[4] P. Dayan and L.F. Abbott. *Theoretical Neuroscience: Computational and Mathematical Modeling of Neural Systems*. MIT Press, 2001.

[5] R. Hahnloser, R. Sarpeshkar, M.A. Mahowald, R.J. Douglas, and S. Seung. Digital selection and analog amplification co-exist in an electronic circuit inspired by neocortex. *Nature*, 405(6789):947–951, 2000.

[6] R.J. Douglas, M.A. Mahowald, and K.A.C. Martin. Hybrid analog-digital architectures for neuromorphic systems. In *Proc. IEEE World Congress on Computational Intelligence*, volume 3, pages 1848–1853. IEEE, 1994.

[7] G. Indiveri. Synaptic plasticity and spike-based computation in VLSI networks of integrate-and-fire neurons. *Neural Information Processing - Letters and Reviews*, 2007. (In press).

[8] G. Indiveri, E. Chicca, and R. Douglas. A VLSI array of low-power spiking neurons and bistable synapses with spike–timing dependent plasticity. *IEEE Transactions on Neural Networks*, 17(1):211–221, Jan 2006.

[9] C. Bartolozzi and G. Indiveri. Synaptic dynamics in analog VLSI. *Neural Computation*, 19:2581–2603, Oct 2007.

[10] Winfried Lohmiller and Jean-Jacques E. Slotine. On contraction analysis for non-linear systems. *Automatica*, 34(6):683–696, 1998.

[11] B. Ermentrout. Reduction of conductance-based models with slow synapses to neural nets. *Neural Computation*, 6:679–695, 1994.

[12] Jean-Jacques E. Slotine. Modular stability tools for distributed computation and control. *International J. of Adaptive Control and Signal Processing*, 17(6):397–416, 2003.

[13] Roger A. Horn and Charles R. Johnson. *Matrix Analysis*. Cambridge University Press, 1985.

[14] Winfried Lohmiller and Jean-Jacques E. Soltine. Nonlinear process control using contraction theory. *A.I.Ch.E. Journal*, March 2000.

[15] S. H. Strogatz. *Nonlinear Dynamics and Chaos*. Perseus Books, 1994.

[16] O. Faugeras and J.-J. Slotine. Synchronization in neural fields. 2007.

[17] Wei Wang and Jean-Jacques E. Slotine. On partial contraction analysis for coupled nonlinear oscillators. *Biological Cybernetics*, 92(1):38–53, 2005.

[18] C. Bartolozzi, S. Mitra, and G. Indiveri. An ultra low power current–mode filter for neuromorphic systems and biomedical signal processing. In *IEEE Proceedings on Biomedical Circuits and Systems (BioCAS06)*, pages 130–133, 2006.

[19] E. Chicca, G. Indiveri, and R.J. Douglas. Context dependent amplification of both rate and event-correlation in a VLSI network of spiking neurons. In B. Schölkopf, J.C. Platt, and T. Hofmann, editors, *Advances in Neural Information Processing Systems 19*, Cambridge, MA, Dec 2007. Neural Information Processing Systems Foundation, MIT Press. (In press).

[20] S. Fusi and M. Mattia. Collective behavior of networks with linear (VLSI) integrate and fire neurons. *Neural Computation*, 11:633–52, 1999.

[21] H. Sebastion Seung Richard H. R. Hahnloser and Jean-Jacques Slotine. Permitted and forbidden sets in symmetric threshold-linear networks. *Neural Computation*, 15:621–638, 2003.

[22] E. Chicca. *A Neuromorphic VLSI System for Modeling Spike–Based Cooperative Competitive Neural Networks*. PhD thesis, ETH Zürich, Zürich, Switzerland, April 2006.

